# Kernel Feature Spaces and Nonlinear Blind Source Separation

**Stefan Harmeling**[1]*, **Andreas Ziehe**[1], **Motoaki Kawanabe**[1], **Klaus-Robert Müller**[1,2]

[1]Fraunhofer FIRST.IDA, Kekuléstr. 7, 12489 Berlin, Germany
[2]University of Potsdam, Department of Computer Science,
August-Bebel-Strasse 89, 14482 Potsdam, Germany
{harmeli,ziehe,kawanabe,klaus}@first.fhg.de

## Abstract

In kernel based learning the data is mapped to a kernel feature space of a dimension that corresponds to the number of training data points. In practice, however, the data forms a smaller submanifold in feature space, a fact that has been used e.g. by reduced set techniques for SVMs. We propose a new mathematical construction that permits to adapt to the intrinsic dimension and to find an orthonormal basis of this submanifold. In doing so, computations get much simpler and more important our theoretical framework allows to derive elegant kernelized blind source separation (BSS) algorithms for arbitrary invertible nonlinear mixings. Experiments demonstrate the good performance and high computational efficiency of our kTDSEP algorithm for the problem of nonlinear BSS.

## 1 Introduction

In a widespread area of applications kernel based learning machines, e.g. Support Vector Machines (e.g. [19, 6]) give excellent solutions. This holds both for problems of supervised and unsupervised learning (e.g. [3, 16, 12]). The general idea is to map the data $\mathbf{x}_i$ ($i = 1, \ldots, T$) into some kernel feature space $\mathcal{F}$ by some mapping $\Phi : \Re^n \to \mathcal{F}$. Performing a simple linear algorithm in $\mathcal{F}$, then corresponds to a nonlinear algorithm in input space. Essential ingredients to kernel based learning are (a) VC theory that can provide a relation between the complexity of the function class in use and the generalization error and (b) the famous kernel trick

$$\mathbf{k}(\mathbf{x}, \mathbf{y}) = \Phi(\mathbf{x}) \cdot \Phi(\mathbf{y}), \tag{1}$$

which allows to efficiently compute scalar products. This trick is essential if e.g. $\mathcal{F}$ is an infinite dimensional space. Note that even though $\mathcal{F}$ might be infinite dimensional the subspace where the data lies is maximally $T$-dimensional. However, the data typically forms an even smaller subspace in $\mathcal{F}$ (cf. also reduced set methods [15]). In this work we therefore propose a new mathematical construction that allows us to adapt to the intrinsic dimension and to provide an *orthonormal* basis of this submanifold. Furthermore, this makes computations much simpler and provides the basis for a new set of kernelized learning algorithms.

To demonstrate the power of our new framework we will focus on the problem of nonlinear BSS [2, 18, 9, 10, 20, 11, 13, 14, 7, 17, 8] and provide an elegant kernel based algorithm for arbitrary invertible nonlinearities. In nonlinear BSS we observe a mixed signal of the following structure

$$\mathbf{x}_t = \mathbf{f}(\mathbf{s}_t), \tag{2}$$

where $\mathbf{x}_t$ and $\mathbf{s}_t$ are $n \times 1$ column vectors and $\mathbf{f}$ is a possibly nonlinear function from $\Re^n$ to $\Re^n$. In the special case where $\mathbf{f}$ is an $n \times n$ matrix we retrieve standard linear BSS (e.g. [8, 4] and references therein). Nonlinear BSS has so far been only applied to industrial pulp data [8], but a large class of applications where nonlinearities can occur in the mixing process are conceivable, e.g. in the fields of telecommunications, array processing, biomedical data analysis (EEG, MEG, EMG, . . .) and acoustic source separation. Most research has so far centered on post-nonlinear models, i.e.

$$\mathbf{x}_t = \mathbf{f}(\mathbf{A}\mathbf{s}_t), \tag{3}$$

where $\mathbf{A}$ is a linear mixing matrix and $\mathbf{f}$ is a post-nonlinearity that operates componentwise. Algorithmic solutions of eq.(3) have used e.g. self-organizing maps [13, 10], extensions of GTM [14], neural networks [2, 11] or ensemble learning [18] to unfold the nonlinearity $\mathbf{f}$. Also a kernel based method was tried on very simple toy signals; however with some stability problems [7]. Note, that all existing methods are of high computational cost and depending on the algorithm are prone to run into local minima. In our contribution to the general invertable nonlinear BSS case we apply a standard BSS technique [21, 1] (that relies on temporal correlations) to mapped signals in feature space (cf. section 3). This is not only mathematically elegant (cf. section 2), but proves to be a remarkably stable and efficient algorithm with high performance, as we will see in the experiments on nonlinear mixtures of toy and speech data (cf. section 4). Finally, a conclusion is given in section 5.

## 2 Theory

**An orthonormal basis for a subspace in $\mathcal{F}$**

In order to establish a linear problem in feature space that corresponds to some nonlinear problem in input space we need to specify how to map inputs $\mathbf{x}_1, \ldots, \mathbf{x}_T \in \Re^n$ into the feature space $\mathcal{F}$ and how to handle its possibly high dimensionality. In addition to the inputs, consider some further points $\mathbf{v}_1, \ldots, \mathbf{v}_d \in \Re^n$ from the same space, that will later generate a basis in $\mathcal{F}$. Alternatively, we could use kernel PCA [16]. However, in this paper we concentrate on a different method. Let us denote the mapped points by $\Phi_{\mathbf{x}} := [\Phi(\mathbf{x}_1) \cdots \Phi(\mathbf{x}_T)]$ and $\Phi_{\mathbf{v}} := [\Phi(\mathbf{v}_1) \cdots \Phi(\mathbf{v}_d)]$. We assume that the columns of $\Phi_{\mathbf{v}}$ constitute a basis of the column space[1] of $\Phi_{\mathbf{x}}$, which we note by

$$\text{span}(\Phi_{\mathbf{v}}) = \text{span}(\Phi_{\mathbf{x}}) \quad \text{and} \quad \text{rank}(\Phi_{\mathbf{v}}) = d. \tag{4}$$

Moreover, $\Phi_{\mathbf{v}}$ being a basis implies that the matrix $\Phi_{\mathbf{v}}^\top \Phi_{\mathbf{v}}$ has full rank and its inverse exists. So, now we can define an orthonormal basis

$$\Xi := \Phi_{\mathbf{v}}(\Phi_{\mathbf{v}}^\top \Phi_{\mathbf{v}})^{-\frac{1}{2}} \tag{5}$$

the column space of which is identical to the column space of $\Phi_{\mathbf{v}}$. Consequently this basis $\Xi$ enables us to parameterize all vectors that lie in the column space of $\Phi_{\mathbf{x}}$ by some vectors in $\Re^d$. For instance for vectors $\sum_{i=1}^{T} \alpha_{\Phi i}\Phi(\mathbf{x}_i)$, which we write more compactly as $\Phi_{\mathbf{x}}\alpha_\Phi$, and $\Phi_{\mathbf{x}}\beta_\Phi$ in the column space of $\Phi_{\mathbf{x}}$ with $\alpha_\Phi$ and $\beta_\Phi$ in $\Re^T$ there exist $\alpha_\Xi$ and $\beta_\Xi$ in $\Re^d$ such that $\Phi_{\mathbf{x}}\alpha_\Phi = \Xi\alpha_\Xi$ and $\Phi_{\mathbf{x}}\beta_\Phi = \Xi\beta_\Xi$. The orthonormality implies

$$\alpha_\Phi^\top \Phi_{\mathbf{x}}^\top \Phi_{\mathbf{x}}\beta_\Phi = \alpha_\Xi^\top \Xi^\top \Xi \beta_\Xi = \alpha_\Xi^\top \beta_\Xi \tag{6}$$

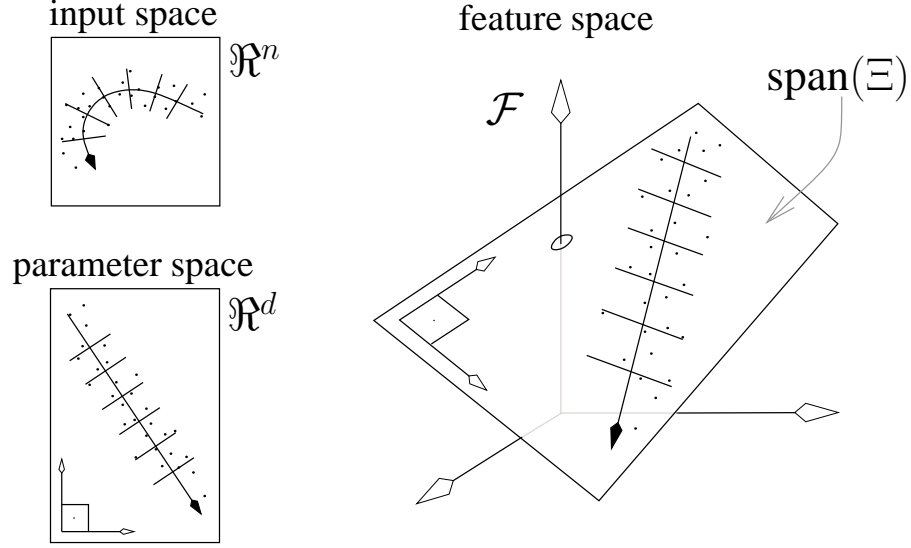

Figure 1: Input data are mapped to some submanifold of $\mathcal{F}$ which is in the span of some $d$-dimensional orthonormal basis $\Xi$. Therefore these mapped points can be parametrized in $\Re^d$. The linear directions in parameter space correspond to nonlinear directions in input space.

which states the *remarkable property* that the dot product of two linear combinations of the columns of $\Phi_{\mathbf{x}}$ in $\mathcal{F}$ coincides with the dot product in $\Re^d$. By construction of $\Xi$ (cf. eq.(5)) the column space of $\Phi_{\mathbf{x}}$ is naturally isomorphic (as vector spaces) to $\Re^d$. Moreover, this isomorphism is compatible with the two involved dot products as was shown in eq.(6). This implies that all properties regarding angles and lengths can be taken back and forth between the column space of $\Phi_{\mathbf{x}}$ and $\Re^d$. The space that is spanned by $\Xi$ is called parameter space. Figure 1 pictures our intuition: Usually kernel methods parameterize the column space of $\Phi_{\mathbf{x}}$ in terms of the mapped patterns $\{\Phi(\mathbf{x}_i)\}$ which effectively corresponds to vectors in $\Re^T$. The orthonormal basis from eq.(5), however enables us to work in $\Re^d$ i.e. in the span of $\Xi$, which is extremely valuable since $d$ depends solely on the kernel function and the dimensionality of the input space. So $d$ is independent of $T$.

**Mapping inputs**

Having established the machinery above, we will now show how to map the input data to the right space. The expressions

$$(\Phi_{\mathbf{v}}^\top \Phi_{\mathbf{v}})_{ij} = \Phi(\mathbf{v}_i)^\top \Phi(\mathbf{v}_j) = \mathbf{k}(\mathbf{v}_i, \mathbf{v}_j) \quad \text{with} \quad i, j = 1 \ldots d$$

are the entries of a real valued $d \times d$ matrix $\Phi_{\mathbf{v}}^\top \Phi_{\mathbf{v}}$ that can be effectively calculated using the kernel trick and by construction of $\mathbf{v}_1, \ldots, \mathbf{v}_d$, it has full rank and is thus invertible. Similarly we get

$$(\Phi_{\mathbf{v}}^\top \Phi_{\mathbf{x}})_{ij} = \Phi(\mathbf{v}_i)^\top \Phi(\mathbf{x}_j) = \mathbf{k}(\mathbf{v}_i, \mathbf{x}_j) \quad \text{with} \quad i = 1 \ldots d, \quad j = 1 \ldots T,$$

which are the entries of the real valued $d \times T$ matrix $\Phi_{\mathbf{v}}^\top \Phi_{\mathbf{x}}$. Using both matrices we compute finally the parameter matrix

$$\Psi_{\mathbf{x}} := \Xi^\top \Phi_{\mathbf{x}} = (\Phi_{\mathbf{v}}^\top \Phi_{\mathbf{v}})^{-\frac{1}{2}} \Phi_{\mathbf{v}}^\top \Phi_{\mathbf{x}} \tag{7}$$

which is also a real valued $d \times T$ matrix; note that $(\Phi_{\mathbf{v}}^{\top} \Phi_{\mathbf{v}})^{-\frac{1}{2}}$ is symmetric. Regarding computational costs, we have to evaluate the kernel function $O(d^2) + O(dT)$ times and eq.(7) requires $O(d^3)$ multiplications; again note that $d$ is much smaller than $T$. Furthermore storage requirements are cheaper as we do not have to hold the full $T \times T$ kernel matrix but only a $d \times T$ matrix. Also, kernel based algorithms often require centering in $\mathcal{F}$, which in our setting is equivalent to centering in $\Re^d$. Fortunately the latter can be done quite cheaply.

**Choosing vectors for the basis in $\mathcal{F}$**

So far we have assumed to have points $\mathbf{v}_1, \ldots, \mathbf{v}_d$ that fulfill eq.(4) and we presented the beneficial properties of our construction. In fact, $\mathbf{v}_1, \ldots, \mathbf{v}_d$ are roughly analogous to a reduced set in the support vector world [15]. Note however that often we can only approximately fulfill eq.(4), i.e.

$$\mathrm{span}(\Phi_{\mathbf{v}}) \approx \mathrm{span}(\Phi_{\mathbf{x}}). \tag{8}$$

In this case we strive for points that provide the best approximation.

Obviously $d$ is finite since it is bounded by $T$, the number of inputs, and by the dimensionality of the feature space. Before formulating the algorithm we define the function $\mathrm{rk}(n)$ for numbers $n$ by the following process: randomly pick $n$ points $\mathbf{v}_1, \ldots, \mathbf{v}_n$ from the inputs and compute the rank of the corresponding $n \times n$ matrix $\Phi_{\mathbf{v}}^{\top} \Phi_{\mathbf{v}}$. Repeating this random sampling process several times (e.g. 100 times) stabilizes this process in practice. Then we denote by $\mathrm{rk}(n)$ the largest achieved rank; note that $\mathrm{rk}(n) \leq n$. Using this definition we can formulate a recipe to find $d$ (the dimension of the subspace of $\mathcal{F}$): (1) start with a large $d$ with $\mathrm{rk}(d) < d$. (2) Decrement $d$ by one as long as $\mathrm{rk}(d) < d$ holds. As soon as we have $\mathrm{rk}(d) = d$ we found the $d$. Choose $\mathbf{v}_1, \ldots, \mathbf{v}_d$ as the vectors that achieve rank $d$. As an alternative to random sampling we have also employed $k$-means clustering with similar results.

## 3  Nonlinear blind source separation

To demonstrate the use of the orthonormal basis in $\mathcal{F}$, we formulate a new nonlinear BSS algorithm based on TDSEP [21]. We start from a set of points $\mathbf{v}_1, \ldots, \mathbf{v}_d$, that are provided by the algorithm from the last section such that eq.(4) holds. Next, we use eq.(7) to compute

$$\Psi_{\mathbf{x}}[t] := \Xi^{\top} \Phi(\mathbf{x}[t]) = (\Phi_{\mathbf{v}}^{\top} \Phi_{\mathbf{v}})^{-\frac{1}{2}} \Phi_{\mathbf{v}}^{\top} \Phi(\mathbf{x}[t]) \quad \in \Re^d.$$

Hereby we have transformed the time signals $\mathbf{x}[t]$ from input space to parameter space signals $\Psi_{\mathbf{x}}[t]$ (cf. Fig.1). Now we apply the usual TDSEP ([21]) that relies on simultaneous diagonalisation techniques [5] to perform linear blind source separation on $\Psi_{\mathbf{x}}[t]$ to obtain $d$ linear directions of separated nonlinear components in input space. This new algorithm is denoted as kTDSEP (kernel TDSEP); in short, kTDSEP is TDSEP on the parameter space defined in Fig.1. A key to the success of our algorithm are the time correlations exploited by TDSEP; intuitively they provide the 'glue' that yields the coherence for the separated signals. Note that for a linear kernel functions the new algorithm performs linear BSS. Therefore linear BSS can be seen as a special case of our algorithm.
Note that common kernel based algorithms which do not use the $d$-dimensional orthonormal basis will run into computational problems. They need to hold and compute with a kernel matrix that is $T \times T$ instead of $d \times T$ with $T \gg d$ in BSS problems. A further problem is that manipulating such a $T \times T$ matrix can easily become unstable. Moreover BSS methods typically become unfeasible for separation problems of dimension $T$.

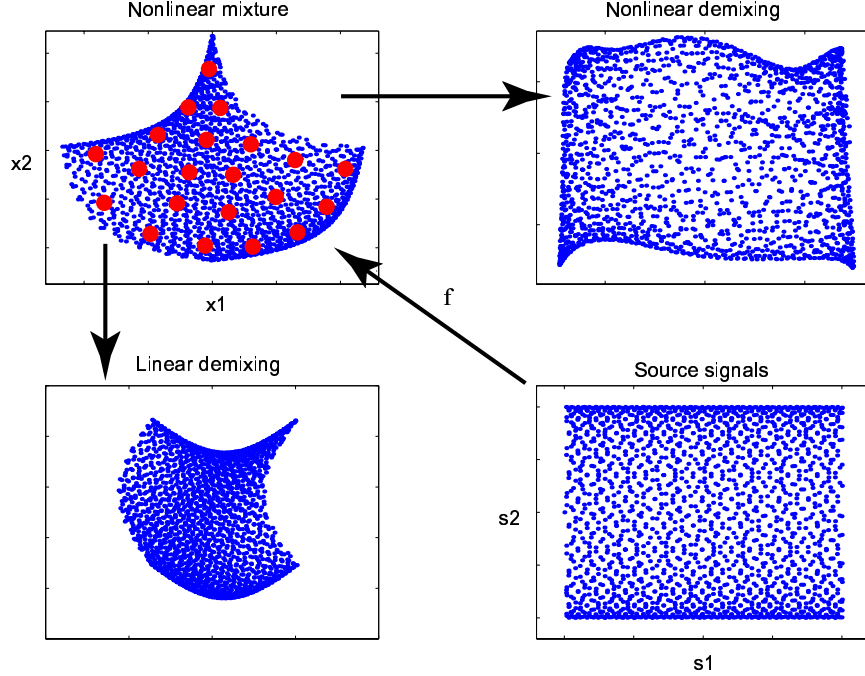

Figure 2: Scatterplot of $x_1$ vs $x_2$ for nonlinear mixing and demixing (upper left and right) and linear demixing and true source signals (lower left and right). Note, that the nonlinear unmixing agrees very nicely with the scatterplot of the true source signal.

## 4 Experiments

In the first experiment the source signals $\mathbf{s}[t] = [s_1[t]\ s_2[t]]^\top$ are a sinusoidal and a sawtooth signal with 2000 samples each. The nonlinearly mixed signals are defined as (cf. Fig.2 upper left panel)

$$
\begin{aligned}
x_1[t] &= \exp(s_1[t]) - \exp(s_2[t]) \\
x_2[t] &= \exp(-s_1[t]) + \exp(-s_2[t]).
\end{aligned}
$$

A dimension $d = 22$ of the manifold in feature space was obtained by kTDSEP using a polynomial kernel $\mathbf{k}(\mathbf{x}, \mathbf{y}) = (\mathbf{x}^\top \mathbf{y} + 1)^6$ by sampling from the inputs. The basis-generating vectors $\mathbf{v}_1, \ldots, \mathbf{v}_{22}$ are shown as big dots in the upper left panel of Figure 2. Applying TDSEP to the 22 dimensional mapped signals $\Psi_{\mathbf{x}}[t]$ we get 22 components in parameter space. A scatter plot with the two components that best match the source signals are shown in the right upper panel of Figure 2. The left lower panel also shows for comparison the two components that we obtained by applying linear TDSEP directly to the mixed signals $\mathbf{x}[t]$. The plots clearly indicate that kTDSEP has unfolded the nonlinearity successfully while the linear demixing algorithm failed.

In a second experiment two speech signals (with 20000 samples, sampling rate 8 kHz) that are nonlinearly mixed by

$$
\begin{aligned}
x_1[t] &= s_1[t] + s_2^3[t] \\
x_2[t] &= s_1^3[t] + \tanh(s_2[t]).
\end{aligned}
$$

This time we used a Gaussian RBF kernel $\mathbf{k}(\mathbf{x}, \mathbf{y}) = \exp(-|\mathbf{x} - \mathbf{y}|^2)$. kTDSEP identified $d = 41$ and used $k$-means clustering to obtain $\mathbf{v}_1, \ldots, \mathbf{v}_{41}$. These points are marked as '+' in the left panel of figure 4. An application of TDSEP to the 41 dimensional parameter

|       | mixture |       | kTDSEP |       | TDSEP |       |
|-------|---------|-------|--------|-------|-------|-------|
|       | $x_1$   | $x_2$ | $u_1$  | $u_2$ | $u_1$ | $u_2$ |
| $s_1$ | 0.56    | 0.72  | 0.89   | 0.07  | 0.09  | 0.72  |
| $s_2$ | 0.63    | 0.46  | 0.04   | 0.86  | 0.31  | 0.55  |

Table 3: Correlation coefficients for the signals shown in Fig.4.

space yields nonlinear components whose projections to the input space are depicted in the right lower panel. We can see that linear TDSEP (right middle panel) failed and that the directions of best matching kTDSEP components closely resemble the sources.

To confirm this visual impression we calculated the correlation coefficients of the kTDSEP and TDSEP solution to the source signals (cf. table 3). Clearly, kTDSEP outperforms the linear TDSEP algorithm, which is of course what one expects.

## 5   Conclusion

Our work has two main contributions. First, we propose a new formulation in the field of kernel based learning methods that allows to construct an orthonormal basis of the subspace of kernel feature space $\mathcal{F}$ where the data lies. This technique establishes a highly useful (scalar product preserving) isomorphism between the image of the data points in $\mathcal{F}$ and a $d$-dimensional space $\Re^d$. Several interesting things follow: we can construct a new set of efficient kernel-based algorithms e.g. a new and eventually more stable variant of kernel PCA [16]. Moreover, we can acquire knowledge about the intrinsic dimension of the data manifold in $\mathcal{F}$ from the learning process.

Second, using our new formulation we tackle the problem of nonlinear BSS from the viewpoint of kernel based learning. The proposed kTDSEP algorithm allows to unmix arbitrary invertible nonlinear mixtures with low computational costs. Note, that the important ingredients are the *temporal correlations* of the source signals used by TDSEP. Experiments on toy and speech signals underline that an elegant solution has been found to a challenging problem.

Applications where nonlinearly mixed signals can occur, are found e.g. in the fields of telecommunications, array processing, biomedical data analysis (EEG, MEG, EMG, ...) and acoustic source separation. In fact, our algorithm would allow to provide a software-based correction of sensors that have a nonlinear characteristics e.g. due to manufacturing errors. Clearly kTDSEP is only one algorithm that can perform nonlinear BSS; kernelizing other ICA algorithms can be done following our reasoning.

**Acknowledgements** The authors thank Benjamin Blankertz, Gunnar Rätsch, Sebastian Mika for valuable discussions. This work was partly supported by the EU project (IST-1999-14190 – BLISS) and DFG (JA 379/9-1, MU 987/1-1).

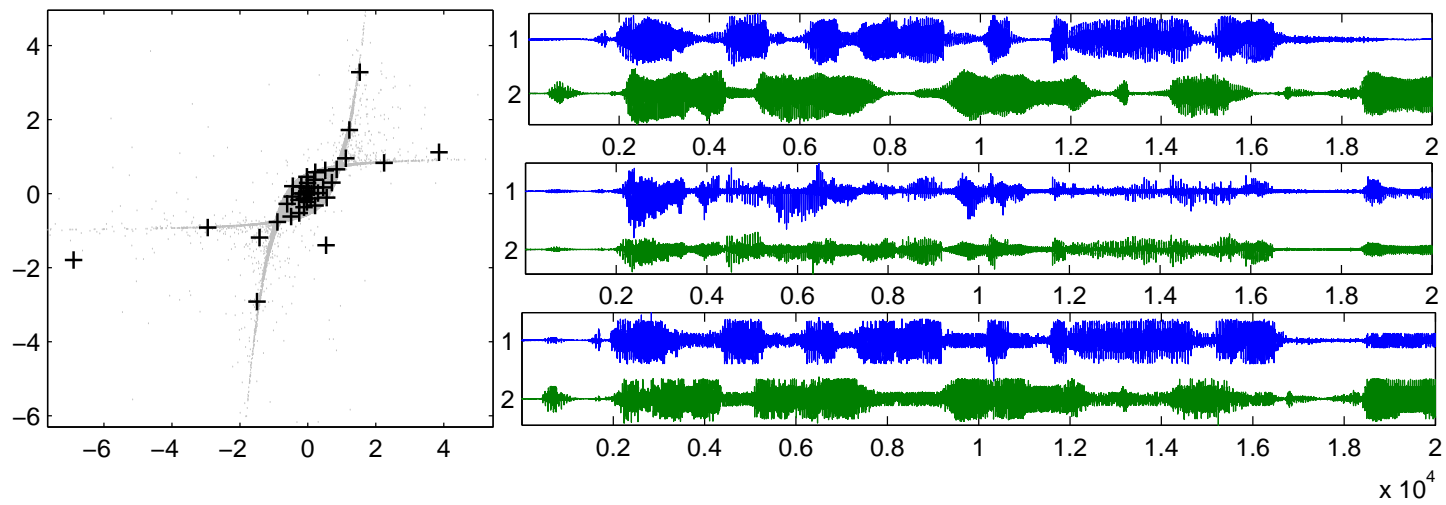

Figure 4: A highly nonlinear mixture of two speech signals: Scatterplot of $x_1$ vs $x_2$ and the waveforms of the true source signals (upper panel) in comparison to the best matching linear and nonlinear separation results are shown in the middle and lower panel, respectively.

## Footnotes

*To whom correspondence should be addressed.

[1]The column space of $\Phi_{\mathbf{x}}$ is the space that is spanned by the column vectors of $\Phi_{\mathbf{x}}$, written span($\Phi_{\mathbf{x}}$).

## References

[1] A. Belouchrani, K. Abed Meraim, J.-F. Cardoso, and E. Moulines. A blind source separation technique based on second order statistics. *IEEE Trans. on Signal Processing*, 45(2):434–444, 1997.

[2] G. Burel. Blind separation of sources: a nonlinear neural algorithm. *Neural Networks*, 5(6):937–947, 1992.

[3] C.J.C. Burges. A tutorial on support vector machines for pattern recognition. *Knowledge Discovery and Data Mining*, 2(2):121–167, 1998.

[4] J.-F. Cardoso. Blind signal separation: statistical principles. *Proceedings of the IEEE*, 9(10):2009–2025, 1998.

[5] J.-F. Cardoso and A. Souloumiac. Jacobi angles for simultaneous diagonalization. *SIAM J.Mat.Anal.Appl.*, 17(1):161 ff., 1996.

[6] N. Cristianini and J. Shawe-Taylor. *An Introduction to Support Vector Machines*. Cambridge University Press, Cambridge, UK, 2000.

[7] C. Fyfe and P. L. Lai. ICA using kernel canonical correlation analysis. In *Proc. Int. Workshop on Independent Component Analysis and Blind Signal Separation (ICA2000)*, pages 279–284, Helsinki, Finland, 2000.

[8] A. Hyvarinen, J. Karhunen, and E. Oja. *Independent Component Analysis*. Wiley, 2001.

[9] T.-W. Lee, B.U. Koehler, and R. Orglmeister. Blind source separation of nonlinear mixing models. In *Neural Networks for Signal Processing VII*, pages 406–415. IEEE Press, 1997.

[10] J. K. Lin, D. G. Grier, and J. D. Cowan. Faithful representation of separable distributions. *Neural Computation*, 9(6):1305–1320, 1997.

[11] G. Marques and L. Almeida. Separation of nonlinear mixtures using pattern repulsion. In *Proc. Int. Workshop on Independent Component Analysis and Signal Separation (ICA'99)*, pages 277–282, Aussois, France, 1999.

[12] K.-R. Müller, S. Mika, G. Rätsch, K. Tsuda, and B. Schölkopf. An introduction to kernel-based learning algorithms. *IEEE Transactions on Neural Networks*, 12(2):181–201, 2001.

[13] P. Pajunen, A. Hyvärinen, and J. Karhunen. Nonlinear blind source separation by self-organizing maps. In *Proc. Int. Conf. on Neural Information Processing*, pages 1207–1210, Hong Kong, 1996.

[14] P. Pajunen and J. Karhunen. A maximum likelihood approach to nonlinear blind source separation. In *Proceedings of the 1997 Int. Conf. on Artificial Neural Networks (ICANN'97)*, pages 541–546, Lausanne, Switzerland, 1997.

[15] B. Schölkopf, S. Mika, C.J.C. Burges, P. Knirsch, K.-R. Müller, G. Rätsch, and A.J. Smola. Input space vs. feature space in kernel-based methods. *IEEE Transactions on Neural Networks*, 10(5):1000–1017, September 1999.

[16] B. Schölkopf, A.J. Smola, and K.-R. Müller. Nonlinear component analysis as a kernel eigenvalue problem. *Neural Computation*, 10:1299–1319, 1998.

[17] A. Taleb and C. Jutten. Source separation in post-nonlinear mixtures. *IEEE Trans. on Signal Processing*, 47(10):2807–2820, 1999.

[18] H. Valpola, X. Giannakopoulos, A. Honkela, and J. Karhunen. Nonlinear independent component analysis using ensemble learning: Experiments and discussion. In *Proc. Int. Workshop on Independent Component Analysis and Blind Signal Separation (ICA2000)*, pages 351–356, Helsinki, Finland, 2000.

[19] V.N. Vapnik. *The nature of statistical learning theory*. Springer Verlag, New York, 1995.

[20] H. H. Yang, S.-I. Amari, and A. Cichocki. Information-theoretic approach to blind separation of sources in non-linear mixture. *Signal Processing*, 64(3):291–300, 1998.

[21] A. Ziehe and K.-R. Müller. TDSEP—an efficient algorithm for blind separation using time structure. In *Proc. Int. Conf. on Artificial Neural Networks (ICANN'98)*, pages 675–680, Skövde, Sweden, 1998.
